# Explanation-Based Neural Network Learning for Robot Control

**Tom M. Mitchell**
School of Computer Science
Carnegie Mellon University
Pittsburgh, PA 15213
E-mail: mitchell@cs.cmu.edu

**Sebastian B. Thrun**
University of Bonn
Institut für Informatik III
Römerstr. 164, D-5300 Bonn, Germany
thrun@uran.informatik.uni-bonn.de

## Abstract

How can artificial neural nets generalize better from fewer examples? In order to generalize successfully, neural network learning methods typically require large training data sets. We introduce a neural network learning method that generalizes rationally from many fewer data points, relying instead on prior knowledge encoded in previously learned neural networks. For example, in robot control learning tasks reported here, previously learned networks that model the effects of robot actions are used to guide subsequent learning of robot control functions. For each observed training example of the target function (e.g. the robot control policy), the learner *explains* the observed example in terms of its prior knowledge, then *analyzes* this explanation to infer additional information about the shape, or slope, of the target function. This shape knowledge is used to bias generalization when learning the target function. Results are presented applying this approach to a simulated robot task based on reinforcement learning.

## 1 Introduction

Neural network learning methods generalize from observed training data to new cases based on an inductive bias that is similar to smoothly interpolating between observed training points. Theoretical results [Valiant, 1984], [Baum and Haussler, 1989] on learnability, as well as practical experience, show that such purely inductive methods require significantly larger training data sets to learn functions of increasing complexity. This paper introduces explanation-based neural network learning (EBNN), a method that generalizes successfully from fewer training examples, relying instead on prior knowledge encoded in previously learned neural networks.

EBNN is a neural network analogue to symbolic explanation-based learning methods (EBL) [DeJong and Mooney, 1986], [Mitchell *et al.*, 1986]. Symbolic EBL methods generalize based upon pre-specified domain knowledge represented by collections of symbolic rules.

For example, in the task of learning general rules for robot control EBL can use prior knowledge about the effects of robot actions to analytically generalize from specific training examples of successful control actions. This is achieved by a. *observing* a sequence of states and actions leading to some goal, b. *explaining* (i.e., post-facto predicting) the outcome of this sequence using the domain theory, then c. *analyzing* this explanation in order to determine which features of the initial state are relevant to achieving the goal of the sequence, and which are not. In previous approaches to EBL, the initial domain knowledge has been represented symbolically, typically by propositional rules or horn clauses, and has typically been assumed to be complete and correct.

## 2    EBNN: Integrating inductive and analytical learning

EBNN extends explanation-based learning to cover situations in which prior knowledge (also called the domain theory) is approximate and is itself learned from scratch. In EBNN, this domain theory is represented by real-valued neural networks. By using neural network representations, it becomes possible to learn the domain theory using training algorithms such as the Backpropagation algorithm [Rumelhart *et al.*, 1986]. In the robot domains addressed in this paper, such domain theory networks correspond to *action models*, i.e., networks that model the effect of actions on the state of the world $M: s \times a \longrightarrow s'$ (here $a$ denotes an action, $s$ a state, and $s'$ the successor state). This domain theory is used by EBNN to bias the learning of the robot control function. Because the action models may be only approximately correct, we require that EBNN be *robust* with respect to severe errors in the domain theory.

The remainder of this section describes the EBNN learning algorithm. Assume that the robot agent's action space is discrete, and that its domain knowledge is represented by a collection of pre-trained action models $M_i: s \longrightarrow s'$, one for each discrete action $i$. The learning task of the robot is to learn a policy for action selection that maximizes the *reward*, denoted by $R$, which defines the task. More specifically, the agent has to learn an *evaluation function* $Q(s, a)$, which measures the *cumulative future expected reward* when action $a$ is executed at state $s$. Once learned, the function $Q(s, a)$ allows the agent to select actions that maximize the reward $R$ (greedy policy). Hence learning control reduces to learning the evaluation function $Q$.[1]

How can the agent use its previously learned action models to focus its learning of $Q$? To illustrate, consider the episode shown in Figure 1. The EBNN learning algorithm for learning the target function $Q$ consists of two components, an *inductive learning component* and an *analytical learning component*.

### 2.1    The inductive component of EBNN

The observed episode is used by the agent to construct training examples, denoted by $\hat{Q}$, for the evaluation function $Q$:

$$\hat{Q}(s_1, a_1) := R \qquad \hat{Q}(s_2, a_2) := R \qquad \hat{Q}(s_3, a_3) := R$$

$Q$ could for example be realized by a monolithic neural network, or by a collection of networks trained with the Backpropagation training procedure. As observed training episodes are accumulated, $Q$ will become increasingly accurate. Such pure inductive learning typ-

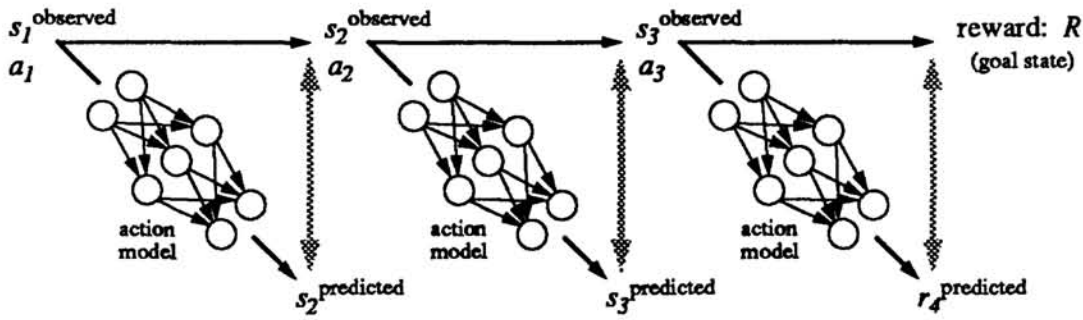

**Figure 1: Episode:** Starting with the initial state $s_1$, the action sequence $a_1$, $a_2$, $a_3$ was observed to produce the final reward $R$. The domain knowledge represented by neural network action models is used to post-facto predict and analyze each step of the observed episode.

ically requires large amounts of training data (which will be costly in the case of robot learning).

## 2.2 The analytical component of EBNN

In EBNN, the agent exploits its domain knowledge to extract additional shape knowledge about the target function $Q$, to speed convergence and reduce the number of training examples required. This shape knowledge, represented by the estimated *slope* of the target function $Q$, is then used to guide the generalization process. More specifically, EBNN combines the above inductive learning component with an analytical learning component that performs the following three steps for each observed training episode:

1. **Explain:** Post-facto predict the observed episode (states and final reward), using the action models $M_i$ (c.f. Fig. 1). Note that there may be a deviation between predicted and observed states, since the domain knowledge is only approximately correct.

2. **Analyze:** Analyze the explanation to estimate the *slope* of the target function for each observed state-action pair $\langle s_k, a_k \rangle$ ($k = 1..3$), i.e., extract the *derivative* of the final reward $R$ with respect to the features of the states $s_k$, according to the action models $M_i$. For instance, consider the explanation of the episode shown in Fig. 1. The domain theory networks $M_i$ represent differentiable functions. Therefore it is possible to extract the derivative of the final reward $R$ with respect to the preceding state $s_3$, denoted by $\nabla_{s_3} R$. Using the chain rule of differentiation, the derivatives of the final reward $R$ with respect to all states $s_k$ can be extracted. These derivatives $\nabla_{s_k} R$ describe the dependence of the final reward upon features of the previous states. They provide the *target slopes*, denoted by $\widehat{\nabla_{s_k} Q}$, for the target function $Q$:

$$\widehat{\nabla_{s_3} Q}(s_3, a_3) \;=\; \nabla_{s_3} R \;=\; \frac{\partial M_{a_3}(s_3)}{\partial s_3}$$

$$\widehat{\nabla_{s_2} Q}(s_2, a_2) \;=\; \nabla_{s_2} R \;=\; \frac{\partial M_{a_3}(s_3)}{\partial s_3} \cdot \frac{\partial M_{a_2}(s_2)}{\partial s_2}$$

$$\widehat{\nabla_{s_1} Q}(s_1, a_1) \;=\; \nabla_{s_1} R \;=\; \frac{\partial M_{a_3}(s_3)}{\partial s_3} \cdot \frac{\partial M_{a_2}(s_2)}{\partial s_2} \cdot \frac{\partial M_{a_1}(s_1)}{\partial s_1}$$

3. **Learn:** Update the learned target function to better fit both the target values and target slopes. Fig. 2 illustrates training information extracted by both the inductive (values) and the analytical (slopes) components of EBNN. Assume that the "true" $Q$-function

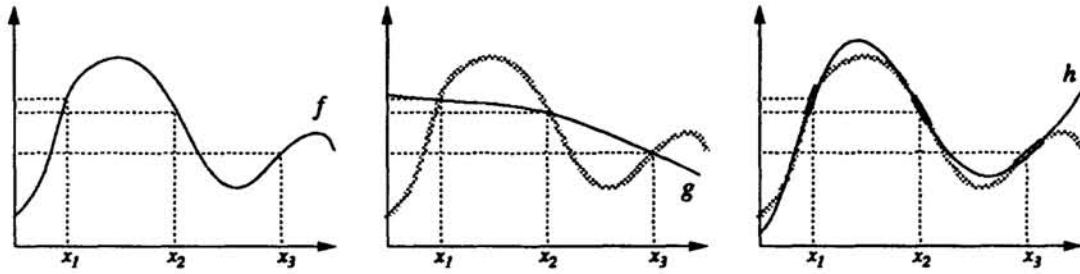

**Figure 2: Fitting slopes:** Let $f$ be a target function for which three examples $\langle x_1, f(x_1)\rangle$, $\langle x_2, f(x_2)\rangle$, and $\langle x_3, f(x_3)\rangle$ are known. Based on these points the learner might generate the hypothesis $g$. If the slopes are also known, the learner can do much better: $h$.

is shown in Fig. 2a, and that three training instances at $x_1$, $x_2$ and $x_3$ are given. When only values are used for learning, i.e., as in standard inductive learning, the learner might conclude the hypothesis g depicted in Fig. 2b. If the slopes are known as well, the learner can better estimate the target function (Fig. 2c). From this example it is clear that the analysis in EBNN may reduce the need for training data, provided that the estimated slopes extracted from the explanations are sufficiently accurate.

In EBNN, the function $Q$ is learned by a real-valued function approximator that fits both the target values and target slopes. If this approximator is a neural network, an extended version of the Backpropagation algorithm can be employed to fit these slope constraints as well, as originally shown by [Simard *et al.*, 1992]. Their algorithm *"Tangent Prop"* extends the Backpropagation error function by a second term measuring the mean square error of the slopes. Gradient descent in slope space is then combined with Backpropagation to minimize both error functions. In the experiments reported here, however, we used an instance-based function approximation technique described in Sect. 3.

### 2.3 Accommodating imperfect domain theories

Notice that the slopes extracted from explanations will be only approximately correct, since they are derived from the approximate action models $M_i$. If this domain knowledge is weak, the slopes can be arbitrarily poor, which may mislead generalization.

EBNN reduces this undesired effect by estimating the *accuracy* of the extracted slopes and weighting the analytical component of learning by these estimated slope accuracies. Generally speaking, *the accuracy of slopes is estimated by the prediction accuracy of the explanation* (this heuristic has been named LOB*). More specifically, each time the domain theory is used to post-facto predict a state $s_{k+1}$, its prediction $s_{k+1}^{\text{predicted}}$ may deviate from the observed state $s_{k+1}^{\text{observed}}$. Hence the 1-step prediction accuracy at state $s_k$, denoted by $c_1(i)$, is defined as 1 minus the normalized prediction error:

$$c_1(i) := 1 - \frac{\|s_{k+1}^{\text{predicted}} - s_{k+1}^{\text{observed}}\|}{\text{max\_prediction\_error}}$$

For a given episode we define the $n$-step accuracy $c_n(i)$ as the product of the 1-step accuracies in the next $n$ steps. The $n$-step accuracy, which measures the accuracy of the derived slopes $n$ steps away from the end of the episode, posseses three desireable properties: a. It is 1 if the learned domain theory is perfectly correct, b. it decreases monotonically as the length of the chain of inferences increases, and c. it is bounded below by 0. The $n$-step accuracy is used to determine the ratio with which the analytical and inductive components

are weighted when learning the target concept. If an observation is $n$ steps away from the end of the episode, the analytically derived training information (slopes) is weighted by the $n$-step accuracy times the weight of the inductive component (values). Although the experimental results reported in section 3 are promising, the generality of this approach is an open question, due to the heuristic nature of the assumption LOB*.

### 2.4   EBNN and Reinforcement Learning

To make EBNN applicable to robot learning, we extend it here to a more sophisticated scheme for learning the evaluation function $Q$, namely Watkins' $Q$-Learning [Watkins, 1989] combined with Sutton's temporal difference methods [Sutton, 1988]. The reason for doing so is the *problem of suboptimal action choices* in robot learning: Robots must explore their environment, i.e., they must select non-optimal actions. Such non-optimal actions can have a negative impact on the final reward of an episode which results in both underestimating target values and misleading slope estimates.

Watkins' $Q$-Learning [Watkins, 1989] permits non-optimal actions during the course of learning $Q$. In his algorithm targets for $Q$ are constructed *recursively*, based on the maximum possible $Q$-value at the next state:[2]

$$\widehat{Q}(s_k, a_k) = \begin{cases} R & \text{if } k \text{ is the final step and } R \text{ final reward} \\ \gamma \max_{a \text{ action}} Q(s_{k+1}, a) & \text{otherwise} \end{cases}$$

Here $\gamma$ ($0 \le \gamma \le 1$) is a *discount factor* that discounts reward over time, which is commonly used for minimizing the number of actions. Sutton's TD($\lambda$) [Sutton, 1988] can be used to combine both Watkins' $Q$-Learning and the non-recursive $Q$-estimation scheme underlying the previous section. Here the parameter $\lambda$ ($0 \le \lambda \le 1$) determines the ratio between recursive and non-recursive components:

$$\widehat{Q}(s_k, a_k) = \begin{cases} R & \text{if } k \text{ final step} \\ (1-\lambda)\,\gamma\,\max_a Q(s_{k+1}, a) + \lambda\,\gamma\,\widehat{Q}(s_{k+1}, a_{k+1}) & \text{otherwise} \end{cases} \quad (1)$$

Eq. (1) describes the extended inductive component of the EBNN learning algorithm. The extension of the analytical component in EBNN is straightforward. Slopes are extracted via the *derivative* of Eq. (1), which is computed via the derivative of both the models $M_i$ and the derivative of $Q$.

$$\widehat{\nabla_{s_k}} Q(s_k, a_k) = \begin{cases} \dfrac{\partial M_{a_k}(s_k)}{\partial s_k} & \text{if } k \text{ last step} \\ (1-\lambda)\,\gamma\,\dfrac{\partial Q(s_{k+1}, a)}{\partial s_{k+1}}\,\dfrac{\partial M_{a_k}(s_k)}{\partial s_k} + \lambda\,\gamma\,\widehat{\nabla_{s_{k+1}}} Q(s_{k+1}, a_{k+1}) & \\ & \text{otherwise} \end{cases}$$

## 3   Experimental results

EBNN has been evaluated in a simulated robot navigation domain. The world and the action space are depicted in Fig. 3a&b. The learning task is to find a $Q$ function, for which the greedy policy navigates the agent to its goal location (circle) from arbitrary starting locations, while avoiding collisions with the walls or the obstacle (square). States are

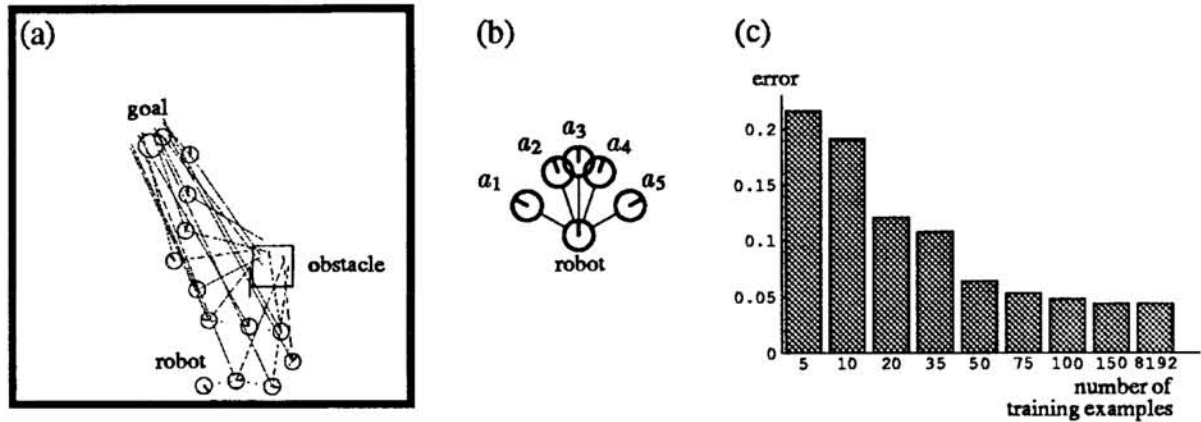

**Figure 3:** a. The simulated robot world. b. Actions. c. The squared generalization error of the domain theory networks decreases monotonically as the amount of training data increases. These nine alternative domain theories were used in the experiments.

described by the local view of the agent, in terms of distances and angles to the center of the goal and to the center of the obstacle. Note that the world is deterministic in these experiments, and that there is no sensor noise.

We applied Watkins' $Q$-Learning and TD($\lambda$) as described in the previous section with $\lambda=0.7$ and a discount factor $\gamma=0.8$. Each of the five actions was modeled by a separate neural network (12 hidden units) and each had a separate $Q$ evaluation function. The latter functions were represented by a instance-based local approximation technique. In a nutshell, this technique memorizes all training instances and their slopes explicitly, and fits a local quadratic model over the $l=3$ nearest neighbors to the query point, fitting both target values and target slopes. We found empirically that this technique outperformed Tangent Prop in the domain at hand.[3] We also applied an *experience replay* technique proposed by Lin [Lin, 1991] in order to optimally exploit the information given by the observed training episodes.

Fig. 4 shows average performance curves for EBNN using nine different domain theories (action models) trained to different accuracies, with (Fig. 4a) and without (Fig. 4b) taking the $n$-step accuracy of the slopes into account. Fig. 4a shows the main result. It shows clearly that (1) EBNN outperforms purely inductive learning, (2) more accurate domain theories yield better performance than less accurate theories, and (3) EBNN learning degrades gracefully as the accuracy of the domain theory decreases, eventually matching the performance of purely inductive learning. In the limit, as the size of the training data set grows, we expect all methods to converge to the same asymptotic performance.

## 4   Conclusion

Explanation-based neural network learning, compared to purely inductive learning, generalizes more accurately from less training data. It replaces the need for large training data sets by relying instead on a previously learned domain theory, represented by neural networks. In this paper, EBNN has been described and evaluated in terms of robot learning tasks. Because the learned action models $M_i$ are independent of the particular control task (reward function), this knowledge acquired during one task transfers directly to other tasks.

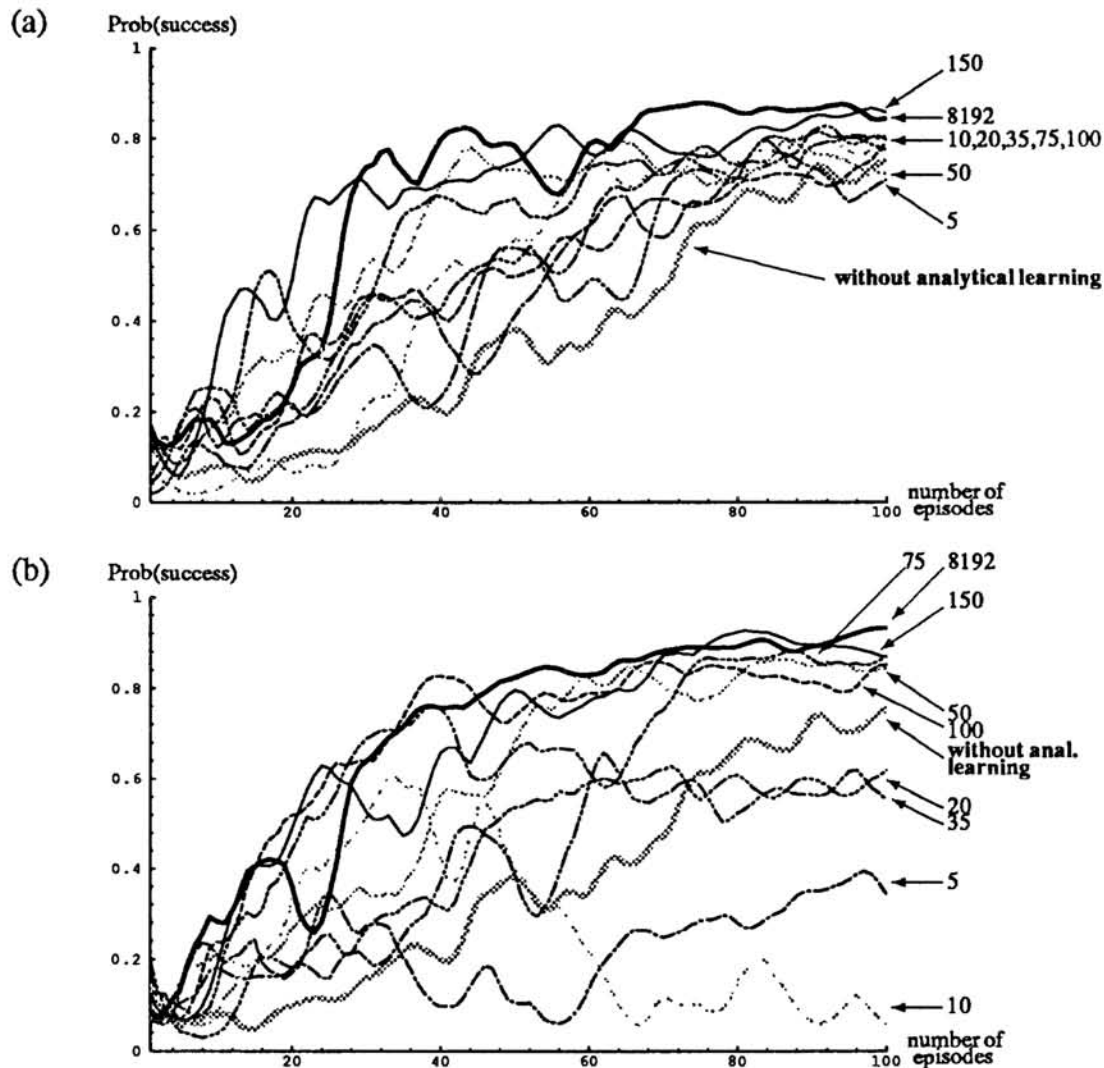

**Figure 4**: How does domain knowledge improve generalization? a. Averaged results for EBNN domain theories of differing accuracies, pre-trained with from 5 to 8 192 training examples for each action model network. In contrast, the bold grey line reflects the learning curve for pure inductive learning, i.e., $Q$-Learning and TD($\lambda$). b. Same experiments, but without weighting the analytical component of EBNN by its accuracy, illustrating the importance of the LOB* heuristic. All curves are averaged over 3 runs and are also locally window-averaged. The performance (vertical axis) is measured on an independent test set of starting positions.

EBNN differs from other approaches to knowledge-based neural network learning, such as Shavlik/Towell's KBANNs [Shavlik and Towell, 1989], in that the domain knowledge and the target function are strictly separated, and that both are learned from scratch. A major difference from other model-based approaches to robot learning, such as Sutton's DYNA architecture [Sutton, 1990] or Jordan/Rumelhart's distal teacher method [Jordan and Rumelhart, 1990], is the ability of EBNN to operate across the spectrum of strong to weak domain theories (using LOB*). EBNN has been found to degrade gracefully as the accuracy of the domain theory decreases.

We have demonstrated the ability of EBNN to transfer knowledge among robot learning tasks. However, there are several open questions which will drive future research, the most significant of which are: a. Can EBNN be extended to real-valued, parameterized

action spaces? So far we assume discrete actions. b. Can EBNN be extended to handle first-order predicate logic, which is common in symbolic approaches to EBL? c. How will EBNN perform in highly stochastic domains? d. Can knowledge other than slopes (such as higher order derivatives) be extracted via explanations? e. Is it feasible to automatically partition/modularize the domain theory as well as the target function, as this is the case with symbolic EBL methods? More research on these issues is warranted.

## Acknowledgments

We thank Ryusuke Masuoka, Long-Ji Lin, the CMU Robot Learning Group, Jude Shavlik, and Mike Jordan for invaluable discussions and suggestions. This research was sponsored in part by the Avionics Lab, Wright Research and Development Center, Aeronautical Systems Division (AFSC), U. S. Air Force, Wright-Patterson AFB, OH 45433-6543 under Contract F33615-90-C-1465, Arpa Order No. 7597 and by a grant from Siemens Corporation.

## Footnotes

[1]This approach to learning a policy is adopted from recent research on *reinforcement learning* [Barto *et al.*, 1991].

[2]In order to simplify the notation, we assume that reward is only received at the end of the episode, and is also modeled by the action models. The extension to more general cases is straightforward.

[3]Note that in a second experiment not reported here, we applied EBNN using neural network representation for $Q$ and Tangent Prop successfully in a real robot domain.

## References

[Barto et al., 1991] Andy G. Barto, Steven J. Bradtke, and Satinder P. Singh. Real-time learning and control using asynchronous dynamic programming. Technical Report COINS 91-57, Department of Computer Science, University of Massachusetts, MA, August 1991.

[Baum and Haussler, 1989] Eric Baum and David Haussler. What size net gives valid generalization? *Neural Computation*, 1(1):151–160, 1989.

[DeJong and Mooney, 1986] Gerald DeJong and Raymond Mooney. Explanation-based learning: An alternative view. *Machine Learning*, 1(2):145–176, 1986.

[Jordan and Rumelhart, 1990] Michael I. Jordan and David E. Rumelhart. Forward models: Supervised learning with a distal teacher. submitted to Cognitive Science, 1990.

[Lin, 1991] Long-Ji Lin. Programming robots using reinforcement learning and teaching. In *Proceedings of AAAI-91*, Menlo Park, CA, July 1991. AAAI Press / The MIT Press.

[Mitchell et al., 1986] Tom M. Mitchell, Rich Keller, and Smadar Kedar-Cabelli. Explanation-based generalization: A unifying view. *Machine Learning*, 1(1):47–80, 1986.

[Pratt, 1993] Lori Y. Pratt. Discriminability-based transfer between neural networks. Same volume.

[Rumelhart et al., 1986] David E. Rumelhart, Geoffrey E. Hinton, and Ronald J. Williams. Learning internal representations by error propagation. In D. E. Rumelhart and J. L. McClelland, editors, *Parallel Distributed Processing. Vol. I + II.* MIT Press, 1986.

[Shavlik and Towell, 1989] Jude W. Shavlik and G.G. Towell. An approach to combining explanation-based and neural learning algorithms. *Connection Science*, 1(3):231–253, 1989.

[Simard et al., 1992] Patrice Simard, Bernard Victorri, Yann LeCun, and John Denker. Tangent prop – a formalism for specifying selected invariances in an adaptive network. In J. E. Moody, S. J. Hanson, and R. P. Lippmann, editors, *Advances in Neural Information Processing Systems 4*, pages 895–903, San Mateo, CA, 1992. Morgan Kaufmann.

[Sutton, 1988] Richard S. Sutton. Learning to predict by the methods of temporal differences. *Machine Learning*, 3, 1988.

[Sutton, 1990] Richard S. Sutton. Integrated architectures for learning, planning, and reacting based on approximating dynamic programming. In *Proceedings of the Seventh International Conference on Machine Learning, June 1990*, pages 216–224, 1990.

[Valiant, 1984] Leslie G. Valiant. A theory of the learnable. *Communications of the ACM*, 27:1134–1142, 1984.

[Watkins, 1989] Chris J. C. H. Watkins. *Learning from Delayed Rewards*. PhD thesis, King's College, Cambridge, England, 1989.